# Ranking on Data Manifolds

**Dengyong Zhou, Jason Weston, Arthur Gretton,**
**Olivier Bousquet, and Bernhard Schölkopf**
Max Planck Institute for Biological Cybernetics, 72076 Tuebingen, Germany
{*firstname.secondname* }*@tuebingen.mpg.de*

## Abstract

The Google search engine has enjoyed huge success with its web page ranking algorithm, which exploits global, rather than local, hyperlink structure of the web using random walks. Here we propose a simple universal ranking algorithm for data lying in the Euclidean space, such as text or image data. The core idea of our method is to rank the data with respect to the intrinsic manifold structure collectively revealed by a great amount of data. Encouraging experimental results from synthetic, image, and text data illustrate the validity of our method.

## 1 Introduction

The Google search engine [2] accomplishes web page ranking using *PageRank* algorithm, which exploits the global, rather than local, hyperlink structure of the web [1]. Intuitively, it can be thought of as modelling the behavior of a random surfer on the graph of the web, who simply keeps clicking on successive links at random and also periodically jumps to a random page. The web pages are ranked according to the stationary distribution of the random walk. Empirical results show PageRank is superior to the naive ranking method, in which the web pages are simply ranked according to the sum of inbound hyperlinks, and accordingly only the local structure of the web is exploited.

Our interest here is in the situation where the objects to be ranked are represented as vectors in Euclidean space, such as text or image data. Our goal is to rank the data with respect to the intrinsic global manifold structure [6, 7] collectively revealed by a huge amount of data. We believe for many real world data types this should be superior to a local method, which rank data simply by pairwise Euclidean distances or inner products.

Let us consider a toy problem to explain our motivation. We are given a set of points constructed in two moons pattern (Figure 1(a)). A query is given in the upper moon, and the task is to rank the remaining points according to their *relevances* to the query. Intuitively, the relevant degrees of points in the upper moon to the query should decrease along the moon shape. This should also happen for the points in the lower moon. Furthermore, all of the points in the upper moon should be more relevant to the query than the points in the lower moon. If we rank the points with respect to the query simply by Euclidean distance, then the left-most points in the lower moon will be more relevant to the query than the right-most points in the upper moon (Figure 1(b)). Apparently this result is not consistent with our intuition (Figure 1(c)).

We propose a simple universal ranking algorithm, which can exploit the intrinsic manifold

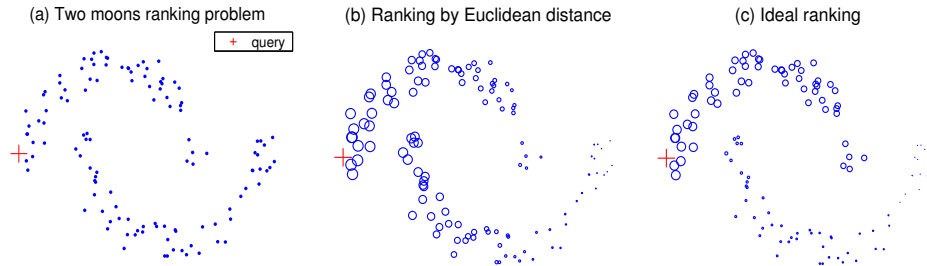

Figure 1: Ranking on the two moons pattern. The marker sizes are proportional to the ranking in the last two figures. (a) toy data set with a single query; (b) ranking by the Euclidean distances; (c) ideal ranking result we hope to obtain.

structure of data. This method is derived from our recent research on semi-supervised learning [8]. In fact the ranking problem can be viewed as an extreme case of semi-supervised learning, in which only positive labeled points are available. An intuitive description of our method is as follows. We first form a weighted network on the data, and assign a positive ranking score to each query and zero to the remaining points which are ranked with respect to the queries. All points then spread their ranking score to their nearby neighbors via the weighted network. The spread process is repeated until a global stable state is achieved, and all points except queries are ranked according to their final ranking scores.

The rest of the paper is organized as follows. Section 2 describes the ranking algorithm in detail. Section 3 discusses the connections with PageRank. Section 4 further introduces a variant of PageRank, which can rank the data with respect to the specific queries. Finally, Section 5 presents experimental results on toy data, on digit image, and on text documents, and Section 6 concludes this paper.

## 2   Algorithm

Given a set of point $\mathcal{X} = \{x_1, ..., x_q, x_{q+1}, ..., x_n\} \subset \mathbb{R}^m$, the first $q$ points are the queries and the rest are the points that we want to rank according to their relevances to the queries.

Let $d : \mathcal{X} \times \mathcal{X} \longrightarrow \mathbb{R}$ denote a metric on $\mathcal{X}$, such as Euclidean distance, which assigns each pair of points $x_i$ and $x_i$ a distance $d(x_i, x_j)$. Let $f : \mathcal{X} \longrightarrow \mathbb{R}$ denote a ranking function which assigns to each point $x_i$ a ranking value $f_i$. We can view $f$ as a vector $f = [f_1, .., f_n]^T$. We also define a vector $y = [y_1, .., y_n]^T$, in which $y_i = 1$ if $x_i$ is a query, and $y_i = 0$ otherwise. If we have prior knowledge about the confidences of queries, then we can assign different ranking scores to the queries proportional to their respective confidences.

The algorithm is as follows:

1.  Sort the pairwise distances among points in ascending order. Repeat connecting the two points with an edge according the order until a connected graph is obtained.

2.  Form the affinity matrix $W$ defined by $W_{ij} = \exp[-d^2(x_i, x_j)/2\sigma^2]$ if there is an edge linking $x_i$ and $x_j$. Note that $W_{ii} = 0$ because there are no loops in the graph.

3.  Symmetrically normalize $W$ by $S = D^{-1/2}WD^{-1/2}$ in which $D$ is the diagonal matrix with $(i, i)$-element equal to the sum of the $i$-th row of $W$.

4. Iterate $f(t+1) = \alpha S f(t) + (1-\alpha)y$ until convergence, where $\alpha$ is a parameter in $[0, 1)$.

5. Let $f_i^*$ denote the limit of the sequence $\{f_i(t)\}$. Rank each point $x_i$ according its ranking scores $f_i^*$ (largest ranked first).

This iteration algorithm can be understood intuitively. First a connected network is formed in the first step. The network is simply weighted in the second step and the weight is symmetrically normalized in the third step. The normalization in the third step is necessary to prove the algorithm's convergence. In the forth step, all points spread their ranking score to their neighbors via the weighted network. The spread process is repeated until a global stable state is achieved, and in the fifth step the points are ranked according to their final ranking scores. The parameter $\alpha$ specifies the relative contributions to the ranking scores from neighbors and the initial ranking scores. It is worth mentioning that *self-reinforcement* is avoided since the diagonal elements of the affinity matrix are set to zero in the second step. In addition, the information is spread *symmetrically* since $S$ is a symmetric matrix.

About the convergence of this algorithm, we have the following theorem:

**Theorem 1** *The sequence $\{f(t)\}$ converges to $f^* = \beta(I - \alpha S)^{-1}y$, where $\beta = 1 - \alpha$.*

See also [8] for the rigorous proof. Here we only demonstrate how to obtain such a closed form expression. Suppose $f(t)$ converges to $f^*$. Substituting $f^*$ for $f(t+1)$ and $f(t)$ in the iteration equation $f(t+1) = \alpha S f(f) + (1-\alpha)y$, we have

$$f^* = \alpha f^* + (1-\alpha)y, \tag{1}$$

which can be transformed into

$$(I - \alpha S)f^* = (1-\alpha)y.$$

Since $(I - \alpha S)$ is invertible, we have

$$f^* = (1-\alpha)(I - \alpha S)^{-1}y.$$

Clearly, the scaling factor $\beta$ does not make contributions for our ranking task. Hence the closed form is equivalent to

$$f^* = (I - \alpha S)^{-1}y. \tag{2}$$

We can use this closed form to compute the ranking scores of points directly. In large-scale real-world problems, however, we prefer using iteration algorithm. Our experiments show that a few iterations are enough to yield high quality ranking results.

## 3  Connections with Google

Let $G = (V, E)$ denote a directed graph with vertices. Let $W$ denote the $n \times n$ adjacency matrix $W$, in which $W_{ij} = 1$ if there is a link in $E$ from vertex $x_i$ to vertex $x_j$, and $W_{ij} = 0$ otherwise. Note that $W$ is possibly asymmetric. Define a random walk on $G$ determined by the following transition probability matrix

$$P = (1 - \epsilon)U + \epsilon D^{-1}W, \tag{3}$$

where $U$ is the matrix with all entries equal to $1/n$. This can be interpreted as a probability $\epsilon$ of transition to an adjacent vertex, and a probability $1 - \epsilon$ of jumping to any point on the graph uniform randomly. Then the ranking scores over $V$ computed by PageRank is given by the stationary distribution $\pi$ of the random walk.

In our case, we only consider graphs which are undirected and connected. Clearly, $W$ is symmetric in this situation. If we also rank all points without queries using our method, as is done by Google, then we have the following theorem:

**Theorem 2** *For the task of ranking data represented by a connected and undirected graph without queries, $f^*$ and PageRank yield the same ranking list.*

*Proof.* We fist show that the stationary distribution $\pi$ of the random walk used in Google is proportional to the vertex degree if the graph $G$ is undirected and connected. Let **1** denote the $1 \times n$ vector with all entries equal to 1. We have

$$
\begin{aligned}
\mathbf{1}DP &= \mathbf{1}D[(1-\epsilon)U + \epsilon D^{-1}W] = (1-\epsilon)\mathbf{1}DU + \epsilon\mathbf{1}DD^{-1}W \\
&= (1-\epsilon)\mathbf{1}D + \epsilon\mathbf{1}W = (1-\epsilon)\mathbf{1}D + \epsilon\mathbf{1}D = \mathbf{1}D.
\end{aligned}
$$

Let vol $G$ denote the volume of $G$, which is given by the sum of vertex degrees. The stationary distribution is then

$$\pi = \mathbf{1}D/\mathrm{vol}\, G. \tag{4}$$

Note that $\pi$ does not depend on $\epsilon$. Hence $\pi$ is also the the stationary distribution of the random walk determined by the transition probability matrix $D^{-1}W$.

Now we consider the ranking result given by our method in the situation without queries. The iteration equation in the fourth step of our method becomes

$$f(t+1) = Sf(t). \tag{5}$$

A standard result [4] of linear algebra states that if $f(0)$ is a vector not orthogonal to the principal eigenvector, then the sequence $\{f(t)\}$ converges to the principal eigenvector of $S$. Let **1** denotes the $n \times 1$ vector with all entries equal to 1. Then

$$SD^{1/2}\mathbf{1} = D^{-1/2}WD^{-1/2}D^{1/2}\mathbf{1} = D^{-1/2}W\mathbf{1} = D^{-1/2}D\mathbf{1} = D^{1/2}\mathbf{1}.$$

Further, noticing that the maximal eigenvalue of $S$ is 1 [8], we know the principal eigenvector of $S$ is $D^{1/2}\mathbf{1}$. Hence

$$f^* = D^{1/2}\mathbf{1}. \tag{6}$$

Comparing (4) with (6), it is clear that $f^*$ and $\pi$ give the same ranking list. This completes our proof.

## 4 Personalized Google

Although PageRank is designed to rank all points without respect to any query, it is easy to modify for query-based ranking problems. Let $P = D^{-1}W$. The ranking scores given by PageRank are the elements of the convergence solution $\pi^*$ of the iteration equation

$$\pi(t+1) = \alpha P^T \pi(t). \tag{7}$$

By analogy with the algorithm in Section 2, we can add a query term on the right-hand side of (7) for the query-based ranking,

$$\pi(t+1) = \alpha P^T \pi(t) + (1-\alpha)y. \tag{8}$$

This can be viewed as the *personalized* version of PageRank. We can show that the sequence $\{\pi(t)\}$ converges to $\pi^* = (1-\alpha)(I - \alpha P^T)^{-1}y$ as before, which is equivalent to

$$\pi^* = (I - \alpha P^T)^{-1}y. \tag{9}$$

Now let us analyze the connection between (2) and (9). Note that (9) can be transformed into

$$\pi^* = [(D - \alpha W)D^{-1}]^{-1}y = D(D - \alpha W)^{-1}y.$$

In addition, $f^*$ can be represented as

$$f^* = [D^{-1/2}(D - \alpha W)D^{-1/2}]^{-1}y = D^{1/2}(D - \alpha W)^{-1}D^{1/2}y. \tag{10}$$

Hence the main difference between $\pi^*$ and $f^*$ is that in the latter the initial ranking score $y_i$ of each query $x_i$ is weighted with respect to its degree.

The above observation motivates us to propose a more general personalized PageRank algorithm,

$$\pi(t+1) = \alpha P^T \pi(t) + (1-\alpha)D^k y, \tag{11}$$

in which we assign different *importance* to queries with respect to their degree. The closed form of (11) is given by

$$\pi^* = (I - \alpha P^T)^{-1} D^k y. \tag{12}$$

If $k = 0$, (12) is just (9); and if $k = 1$, we have

$$\pi^* = (I - \alpha P^T)^{-1} Dy = D(D - \alpha W)^{-1} Dy,$$

which is almost as same as (10).

We can also use (12) for classification problems without any modification, besides setting the elements of $y$ to 1 or -1 corresponding to the positive or negative classes of the labeled points, and 0 for the unlabeled data. This shows the ranking and classification problems are closely related.

We can do a similar analysis of the relations to Kleinberg's HITS [5], which is another popular web page ranking algorithm. The basic idea of this method is also to iteratively spread the ranking scores via the existing web graph. We omit further discussion of this method due to lack of space.

## 5 Experiments

We validate our method using a toy problem and two real-world domains: image and text. In our following experiments we use the closed form expression in which $\alpha$ is fixed at $0.99$. As a true labeling is known in these problems, i.e. the image and document categories (which is not true in real-world ranking problems), we can compute the ranking error using the Receiver Operator Characteristic (ROC) score [3] to evaluate ranking algorithms. The returned score is between 0 and 1, a score of 1 indicating a perfect ranking.

### 5.1 Toy Problem

In this experiment we considered the toy ranking problem mentioned in the introduction section. The connected graph described in the first step of our algorithm is shown in Figure 2(a). The ranking scores with different time steps: $t = 5, 10, 50, 100$ are shown in Figures 2(b)-(e). Note that the scores on each moon decrease along the moon shape away from the query, and the scores on the moon containing the query point are larger than on the other moon. Ranking by Euclidean distance is shown in Figure 2(f), which fails to capture the two moons structure.

It is worth mentioning that simply ranking the data according to the shortest paths [7] on the graph does not work well. In particular, we draw the reader's attention to the long edge in Figure 2(a) which links the two moons. It appears that shortest paths are sensitive to the small changes in the graph. The robust solution is to assemble all paths between two points, and weight them by a decreasing factor. This is exactly what we have done. Note that the closed form can be expanded as $f^* = \sum_i \alpha^i S^i y$.

### 5.2 Image Ranking

In this experiment we address a task of ranking on the USPS handwritten 16x16 digits dataset. We rank digits from 1 to 6 in our experiments. There are 1269, 929, 824, 852, 716 and 834 examples for each class, for a total of 5424 examples.

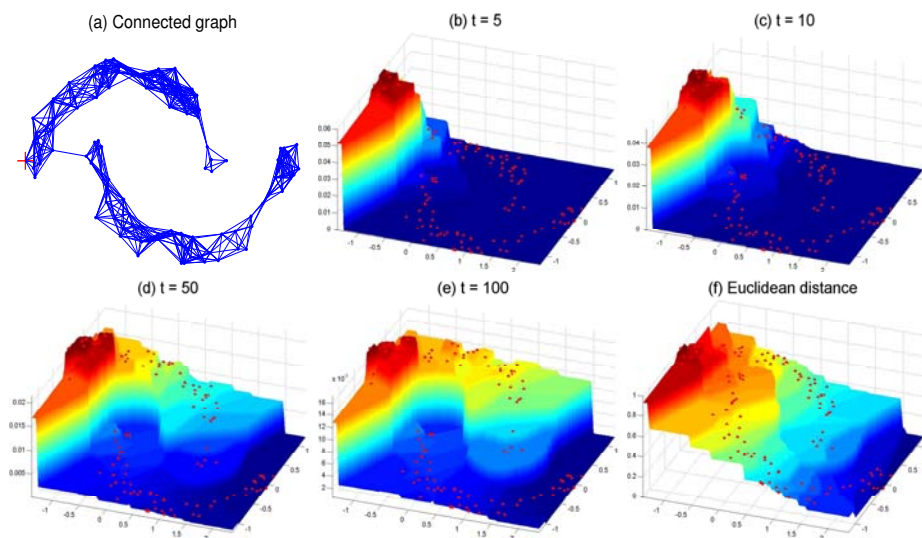

Figure 2: Ranking on the pattern of two moons. (a) connected graph; (b)-(e) ranking with the different time steps: $t = 5, 10, 50, 100$; (f) ranking by Euclidean distance.

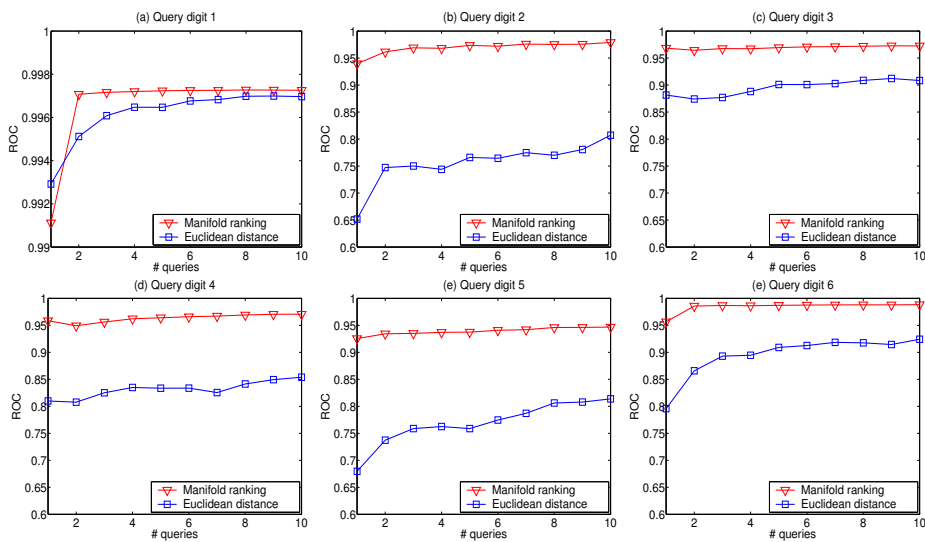

Figure 3: ROC on USPS for queries from digits 1 to 6. Note that this experimental results also provide indirect proof of the intrinsic manifold structure in USPS.

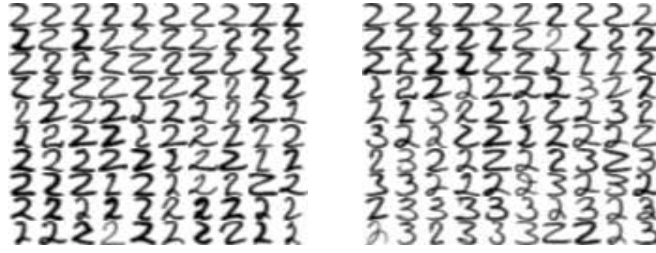

Figure 4: Ranking digits on USPS. The top-left digit in each panel is the query. The left panel shows the top 99 by the manifold ranking; and the right panel shows the top 99 by the Euclidean distance based ranking. Note that there are many more *2*s with knots in the right panel.

We randomly select examples from one class of digits to be the query set over 30 trials, and then rank the remaining digits with respect to these sets. We use a RBF kernel with the width $\sigma = 1.25$ to construct the affinity matrix $W$, but the diagonal elements are set to zero. The Euclidean distance based ranking method is used as the baseline: given a query set $\{x_s\}(s \in S)$, the points $x$ are ranked according to that the highest ranking is given to the point $x$ with the lowest score of $\min_{s \in S}\|x - x_s\|$.

The results, measured as ROC scores, are summarized in Figure 3; each plot corresponds to a different query class, from digit one to six respectively. Our algorithm is comparable to the baseline when a digit *1* is the query. For the other digits, however, our algorithm significantly outperforms the baseline. This experimental result also provides indirect proof of the underlying manifold structure in the USPS digit dataset [6, 7].

The top ranked 99 images obtained by our algorithm and Euclidean distance, with a random digit *2* as the query, are shown in Figure 4. The top-left digit in each panel is the query. Note that there are some *3*s in the right panel. Furthermore, there are many curly *2*s in the right panel, which do not match well with the query: the *2*s in the left panel are *more similar* to the query than the *2*s in the right panel. This subtle superiority makes a great deal of sense in the real-word ranking task, in which users are only interested in very few leading ranking results. The ROC measure is too simple to reflect this subtle superiority however.

### 5.3   Text Ranking

In this experiment, we investigate the task of text ranking using the 20-newsgroups dataset. We choose the topic *rec* which contains *autos, motorcycles, baseball* and *hockey* from the version 20-news-18828.

The articles are processed by the Rainbow software package with the following options: (1) passing all words through the Porter stemmer before counting them; (2) tossing out any token which is on the stoplist of the SMART system; (3) skipping any headers; (4) ignoring words that occur in 5 or fewer documents. No further preprocessing was done. Removing the empty documents, we obtain 3970 document vectors in a 8014-dimensional space. Finally the documents are normalized into TFIDF representation.

We use the ranking method based on normalized inner product as the baseline. The affinity matrix $W$ is also constructed by inner product, i.e. linear kernel. The ROC scores for 100 randomly selected queries for each class are given in Figure 5.

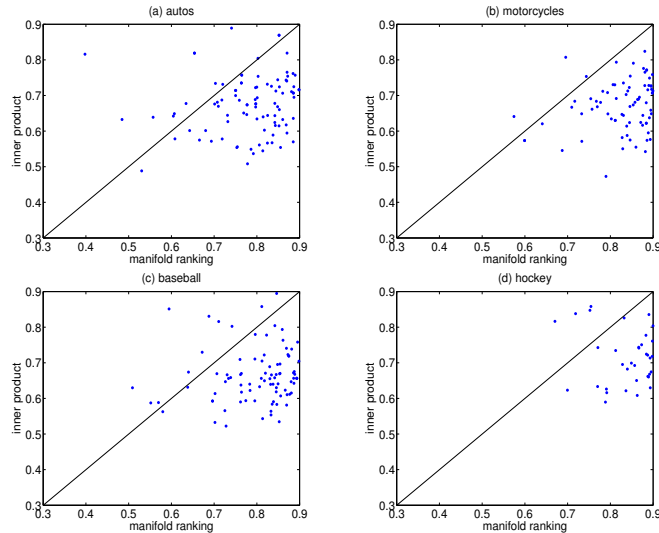

Figure 5: ROC score scatter plots of 100 random queries from the category *autos, motorcycles, baseball* and *hockey* contained in the 20-newsgroups dataset.

## 6  Conclusion

Future research should address model selection. Potentially, if one was given a small labeled set or a query set greater than size 1, one could use standard cross validation techniques. In addition, it may be possible to look to the theory of stability of algorithms to choose appropriate hyperparameters. There are also a number of possible extensions to the approach. For example one could implement an *iterative feedback* framework: as the user specifies positive feedback this can be used to extend the query set and improve the ranking output. Finally, and most importantly, we are interested in applying this algorithm to wide-ranging real-word problems.

## References

[1] R. Albert, H. Jeong, and A. Barabsi. Diameter of the world wide web. *Nature*, 401:130–131, 1999.

[2] S. Brin and L. Page. The anatomy of a large scale hypertextual web search engine. In *Proc. 7th International World Wide Web Conf.*, 1998.

[3] R. Duda, P. Hart, and D. Stork. *Pattern Classification*. Wiley-Interscience, 2nd edition, 2000.

[4] G. H. Golub and C. F. Van Loan. *Matrix Computations*. Johns Hopkins University Press, Baltimore, 1989.

[5] J. Kleinberg. Authoritative sources in a hyperlinked environment. *JACM*, 46(5):604–632, 1999.

[6] S. T. Roweis and L. K. Saul. Nonlinear dimensionality reduction by locally linear embedding. *Science*, 290:2323–2326, 2000.

[7] J. B. Tenenbaum, V. de Silva, and J. C. Langford. Global geometric framework for nonlinear dimensionality reduction. *Science*, 290:2319–2323, 2000.

[8] D. Zhou, O. Bousquet, T. N. Lal, J. Weston, and B. Schölkopf. Learning with local and global consistency. In *18th Annual Conf. on Neural Information Processing Systems*, 2003.
